# Incorporating Second-Order Functional Knowledge for Better Option Pricing

**Charles Dugas, Yoshua Bengio, François Bélisle, Claude Nadeau,\* René Garcia**
*CIRANO*, Montreal, Qc, Canada H3A 2A5
{dugas,bengioy,belislfr,nadeauc}@iro.umontreal.ca
garciar@cirano.qc.ca

## Abstract

Incorporating prior knowledge of a particular task into the architecture of a learning algorithm can greatly improve generalization performance. We study here a case where we know that the function to be learned is non-decreasing in two of its arguments and convex in one of them. For this purpose we propose a class of functions similar to multi-layer neural networks but (1) that has those properties, (2) is a universal approximator of continuous functions with these and other properties. We apply this new class of functions to the task of modeling the price of call options. Experiments show improvements on regressing the price of call options using the new types of function classes that incorporate the *a priori* constraints.

## 1 Introduction

Incorporating *a priori* knowledge of a particular task into a learning algorithm helps reducing the necessary complexity of the learner and generally improves performance, if the incorporated knowledge is relevant to the task and really corresponds to the generating process of the data. In this paper we consider prior knowledge on the positivity of some first and second derivatives of the function to be learned. In particular such constraints have applications to modeling the price of European stock options. Based on the Black-Scholes formula, the price of a call stock option is monotonically increasing in both the "moneyness" and time to maturity of the option, and it is convex in the "moneyness". Section 3 better explains these terms and stock options. For a function $f(x_1, x_2)$ of two real-valued arguments, this corresponds to the following properties:

$$\frac{\partial f}{\partial x_1} \geq 0, \qquad \frac{\partial f}{\partial x_2} \geq 0, \qquad \frac{\partial^2 f}{\partial x_1^2} \geq 0 \qquad (1)$$

The mathematical results of this paper (section 2) are the following: first we introduce a class of one-argument functions (similar to neural networks) that is positive, non-decreasing and convex in its argument, and we show that this class of functions is a universal approximator for positive functions with positive first and second derivatives. Second, in the main theorem, we extend this result to functions of two or more arguments, with some having the convexity property and all having positive first derivative. This result rests on additional properties on cross-derivatives, which we illustrate below for the case of two

arguments:

$$\frac{\partial^2 f}{\partial x_1 \partial x_2} \geq 0, \quad \frac{\partial^3 f}{\partial x_1^2 \partial x_2} \geq 0 \tag{2}$$

Comparative experiments on these new classes of functions were performed on stock option prices, showing some improvements when using these new classes rather than ordinary feedforward neural networks. The improvements appear to be non-stationary but the new class of functions shows the most stable behavior in predicting future prices. The detailed results are presented in section 5.

## 2 Theory

**Definition**
A class of functions $\hat{\mathcal{F}}$ from $\mathbb{R}^n$ to $\mathbb{R}$ is a **universal approximator** for a class of functions $\mathcal{F}$ from $\mathbb{R}^n$ to $\mathbb{R}$ if for any $f \in \mathcal{F}$, any compact domain $D \subset \mathbb{R}^n$, and any positive $\epsilon$, one can find a $\hat{f} \in \hat{\mathcal{F}}$ with $\sup_{x \in D} |f(x) - f(\hat{x})| \leq \epsilon$.

It has already been shown that the class of artificial neural networks with one hidden layer

$$\hat{\mathcal{N}} = \{f(x) = b_0 + \sum_{i=1}^{H} w_i h(b_i + \sum_j v_{ij} x_j)\} \tag{3}$$

e.g. with a sigmoid activation function $h(s) = \frac{1}{1+e^{-s}}$, are universal approximators of continuous functions [1, 2, 5]. The number of hidden units $H$ of the neural network is a hyper-parameter that controls the accuracy of the approximation and it should be chosen to balance the trade-off between accuracy (bias of the class of functions) and variance (due to the finite sample used to estimate the parameters of the model), see also [6].

Since $h$ is monotonically increasing, it is easy to force the first derivatives with respect to $x$ to be positive by forcing the weights to be positive, for example with the exponential function:

$$\hat{\mathcal{N}}_+ = \{f(x) = b_0 + \sum_{i=1}^{H} e^{w_i} h(b_i + \sum_j e^{v_{ij}} x_j)\} \tag{4}$$

because $h'(s) = h(s)(1 - h(s)) > 0$.

Since the sigmoid $h$ has a positive first derivative, its primitive, which we call *softplus*, is convex:

$$\zeta(s) = \log(1 + e^s) \tag{5}$$

i.e., $d\zeta(s)/ds = h(s) = 1/(1 + e^{-s})$. The basic idea of the proposed class of functions $\hat{\mathcal{N}}_{++}$ is to replace the sigmoid of a sum by a product of softplus or sigmoid functions over each of the dimensions (using the softplus over the convex dimensions and the sigmoid over the others):

$$_c\hat{\mathcal{N}}_{++} = \{f(x) = e^{b_0} + \sum_{i=1}^{H} e^{w_i} (\prod_{j=1}^{c} \zeta(b_{ij} + e^{v_{ij}} x_j))(\prod_{j=c+1}^{n} h(b_{ij} + e^{v_{ij}} x_j))\} \tag{6}$$

One can readily check that the first derivatives wrt $x_j$ are positive, and that the second derivatives wrt $x_j$ for $j \leq c$ are positive. However, this class of functions has other properties. Let $(j_1, \cdots, j_m)$ be a set of indices with $1 \leq j_i \leq c$ (convex dimensions), and let $(j'_1, \cdots, j'_p)$ be a set of indices $c + 1 \leq j'_i \leq n$ (the other dimensions), then

$$\frac{\partial^{m+p} f}{\partial x_{j_1} \cdots \partial x_{j_m} \partial x_{j'_1} \cdots x_{j'_p}} \geq 0, \quad \frac{\partial^{2m+p} f}{\partial x_{j_1}^2 \cdots \partial x_{j_m}^2 \partial x_{j'_1} \cdots x_{j'_p}} \geq 0 \tag{7}$$

Note that $m$ or $p$ can be 0, so as special cases we find that $f$ is positive, and that it is monotonically increasing w.r.t. all its inputs, and convex w.r.t. the first $c$ inputs.

## 2.1 Universality of $_c\hat{\mathcal{N}}_{++}$ over $\mathbb{R}$

**Theorem** Within the set $\mathcal{F}_{++}$ of continuous functions from $\mathbb{R}^n$ to $\mathbb{R}$ whose first and second derivatives are non-negative (as specified by equation 7), the class $_c\hat{\mathcal{N}}_{++}$ is a universal approximator.

**Proof**

For lack of space we only show here a sketch of the proof, and only for the case $n = 2$ and $c = 1$ (one convex dimension and one other dimension), but the same principle allows to prove the more general case. Let $f(\vec{x}) \in \mathcal{F}_{++}$ be the function to approximate with a function $g \in {}_1\hat{\mathcal{N}}_{++}$. To perform our approximation we will restrict $g$ to the subset of $_1\hat{\mathcal{N}}_{++}$ where the sigmoid becomes a step function $\theta(x) = I_{x \geq 0}$ and where the softplus becomes the positive part function $x_+ = max(0, x)$. Let $D$ be the compact domain of interest and $\epsilon$ the desired approximation precision. We focus our attention on an axis-aligned rectangle $T$ with lower-left corner $(a_1, b_1)$ and upper right corner $(a_2, b_2)$ such that it is the smallest such rectangle enclosing $D$ and it can be partitionned into squares of length $L$ forming a grid such that the value of $f$ at neighboring grid points does not differ by more than $\epsilon$. The number of square grids on the $x_1$ axis is $N_1$ and the number on the $x_2$ axis is $N_2$. The number of hidden units is $H = (N_1 + 1)(N_2 + 1)$. Let $\vec{x}_{ij} = (x_i, x_j) = (a_1 + iL, b_1 + jL)$ be the grid points, with $i = 0, 1, \ldots, N_1$, $j = 0, 1, \ldots, N_2$. Also, $\vec{x} = (x_1, x_2)$. With $k = i(N_2 + 1) + j$, we recursively build a series of functions $g_k(\vec{x})$ as follows:

$$g_k(\vec{x}) = g_{k-1}(\vec{x}) + \Delta_{ij}(x_1 - x_i + L)_+\theta(x_2 - x_j)$$

with increment

$$\Delta_{ij} = \frac{f(\vec{x}_{ij}) - g_{k-1}(\vec{x})}{L}$$

for $k = 1$ to $H$ and with initial approximation $g_0 = f(a_1, b_1)$. The final approximation is $g(\vec{x}) = g_H(\vec{x})$. It is exact at every single point on the grid and within $\epsilon$ of the true function value anywhere within $D$. To prove this, we need to show that at every step of the recursive procedure, the necessary increment is nonnegative (since it must be equated with $e^{w_k}$). First note that the value of $g_H(\vec{x}_{ij})$ is strictly affected by the set of increments $\Delta_{st}$ for which $s <= i$ and $t <= j$ so that,

$$f(\vec{x}_{ij}) = g_H(\vec{x}_{ij}) = \sum_{s=0}^{i} \sum_{t=0}^{j} \Delta_{st}(i - s + 1)L$$

Isolating $\Delta_{ij}$ and doing some algebra, we get,

$$\Delta_{ij} = \Delta^3_{x_1,x_1,x_2} g_H(\vec{x}_{ij})L^2$$

where $\Delta^3_{x_i,x_j,x_k}$ is the third degree finite difference with respect to arguments $x_i, x_j, x_k$, i.e. $\Delta^3_{x_1,x_1,x_2} f(x_1, x_2) = (\Delta^2_{x_1,x_2} f(x_1, x_2) - \Delta^2_{x_1,x_2} f(x_1 - L, x_2))/L$, where similarly $\Delta^2_{x_1,x_2} f(x_1, x_2) = (\Delta_{x_1} f(x_1, x_2) - \Delta_{x_1} f(x_1, x_2 - L))/L$, and $\Delta_{x_1} f(x_1, x_2) = (f(x_1, x_2) - f(x_1 - L, x_2))/L$. By the mean value theorem, the third degree finite difference is nonnegative if the corresponding third derivative is nonnegative everywhere over the finite interval which is obtained by constraint 7. Finally, the third degree finite difference being nonnegative, the corresponding increment is also nonnegative and this completes the proof.

**Corollary** Within the set of positive continuous functions from $\mathbb{R}$ to $\mathbb{R}$ whose first and second derivatives are non-negative, the class $_1\hat{\mathcal{N}}_{++}$ is a universal approximator.

## 3    Estimating Call Option Prices

An option is a contract between two parties that entitles the buyer to a claim at a future date $T$ that depends on the future price, $S_T$ of an underlying asset whose price at time $t$ is $S_t$. In this paper we consider the very common European call options, in which the value of the claim *at maturity* (time $T$) is $\max(0, S_T - K)$, i.e. if the price is above the *strike price $K$*, then the seller of the option owes $S_T - K$ dollars to the buyer. In the no-arbitrage framework, the call function is believed to be a function of the actual market price of the security ($S_t$), the strike price ($K$), the remaining time to maturity ($\tau = T - t$), the risk free interest rate ($r$), and the volatility of the return ($\sigma$). The challenge is to evaluate the value of the option prior to the expiration date before entering a transaction. The risk free interest rate ($r$) needs to be somehow extracted from the term structure and the volatility ($\sigma$) needs to be forecasted, this latest task being a field of research in itself. We have [3] previously tried to feed in neural networks with estimates of the volatility using historical averages but so far, the gains remained insignificant. We therefore drop these two features and rely on the ones that can be observed: $S_t, K, \tau$. One more important result is that under mild conditions, the call option function is homogeneous of degree one with respect to the strike price and so our final approximation depends on two variables: the moneyness ($M = S_t/K$) and the time to maturity ($\tau$).

$$C_t/K \quad = \quad f(M, \tau) \tag{8}$$

An economic theory yielding to the Black-Scholes formula suggest that $f$ has the properties of (1), so we will evaluate the advantages brought by the function classes of the previous section. However, it is not clear whether the constraint on the cross derivatives that are incorporated in $_1\hat{\mathcal{N}}_{++}$ should or not be present in the true price function. It is known that the Black-Scholes formula does not adequately represent the market pricing of options, but it might still be a useful guide in designing a learning algorithm for option prices.

## 4    Experimental Setup

As a reference model, we use a simple multi-layered perceptron with one hidden layer (eq. 3). We also compare our results with a recently proposed model [4] that closely resembles the Black-Scholes formula for option pricing (i.e. another way to incorporate possibly useful prior knowledge):

$$
\begin{aligned}
y^{BS} \quad = \quad & \alpha + M \cdot \sum_{i=1}^{n_h} \beta_{1,i} \cdot h(\gamma_{i,0} + \gamma_{i,1} \cdot M + \gamma_{i,2} \cdot \tau) \\
+ \quad & e^{-r\tau} \cdot \sum_{i=1}^{n_h} \beta_{2,i} \cdot h(\gamma_{i,3} + \gamma_{i,4} \cdot M + \gamma_{i,5} \cdot \tau).
\end{aligned} \tag{9}
$$

We evaluate two new architectures incorporating some or all of the constraints defined in equation 7.

We used european call option data from 1988 to 1993. A total of 43518 transaction prices on european call options on the S&P500 index were used. In section 5, we report results on 1988 data. In each case, we used the first two quarters of 1988 as a training set (3434 examples), the third quarter as a validation set (1642 examples) for model selection and 4 to 20 quarters as a test sets (each with around 1500 examples) for final generalization error estimation. In tables 1 and 2, we present results for networks with unconstrained weights on the left-hand side, and weights constrained to positive and monotone functions through exponentiation of parameters on the right-hand side. For each model, the number of hidden units varies from one to nine. The mean squared error results reported were obtained as follows: first, we randomly sampled the parameter space 1000 times. We picked the best (lowest training error) model and trained it up to 1000 more times. Repeating this procedure

10 times, we selected and averaged the performance of the best of these 10 models (those with training error no more than 10% worse than the best out of 10). In figure 1, we present tests of the same models on each quarter up to and including 1993 (20 additional test sets) in order to assess the persistence (conversely, the degradation through time) of the trained models.

## 5  Forecasting Results

**Simple Multi-Layered Perceptrons**
Mean Squared Error Results on Call Option Pricing ($\times 10^{-4}$)

| Units | Unconstrained weights | | | | Constrained weights | | | |
|---|---|---|---|---|---|---|---|---|
| | Train | Valid | Test1 | Test2 | Train | Valid | Test1 | Test2 |
| 1 | 2.38 | 1.92 | 2.73 | 6.06 | 2.67 | 2.32 | 3.02 | 3.60 |
| 2 | 1.68 | 1.76 | 1.51 | 5.70 | 2.63 | 2.14 | 3.08 | 3.81 |
| 3 | 1.40 | 1.39 | 1.27 | 27.31 | 2.63 | 2.15 | 3.07 | 3.79 |
| 4 | 1.42 | 1.44 | 1.25 | 27.32 | 2.65 | 2.24 | 3.05 | 3.70 |
| 5 | 1.40 | 1.38 | **1.27** | **30.56** | 2.67 | 2.29 | 3.03 | 3.64 |
| 6 | 1.41 | 1.43 | 1.24 | 33.12 | 2.63 | 2.14 | **3.08** | **3.81** |
| 7 | 1.41 | 1.41 | 1.26 | 33.49 | 2.65 | 2.23 | 3.05 | 3.71 |
| 8 | 1.41 | 1.43 | 1.24 | 39.72 | 2.63 | 2.14 | 3.07 | 3.80 |
| 9 | 1.40 | 1.41 | 1.24 | 38.07 | 2.66 | 2.27 | 3.04 | 3.67 |

**Black-Scholes Similar Networks**
Mean Squared Error Results on Call Option Pricing ($\times 10^{-4}$)

| Units | Unconstrained weights | | | | Constrained weights | | | |
|---|---|---|---|---|---|---|---|---|
| | Train | Valid | Test1 | Test2 | Train | Valid | Test1 | Test2 |
| 1 | 1.54 | 1.58 | 1.40 | 4.70 | 2.49 | 2.17 | 2.78 | 3.61 |
| 2 | 1.42 | 1.42 | 1.27 | 24.53 | 1.90 | 1.71 | 2.05 | 3.19 |
| 3 | 1.40 | 1.41 | 1.24 | 30.83 | 1.88 | 1.73 | 2.00 | 3.72 |
| 4 | 1.40 | 1.39 | **1.27** | **31.43** | 1.85 | 1.70 | 1.96 | 3.15 |
| 5 | 1.40 | 1.40 | 1.25 | 30.82 | 1.87 | 1.70 | 2.01 | 3.51 |
| 6 | 1.41 | 1.42 | 1.25 | 35.77 | 1.89 | 1.70 | 2.04 | 3.19 |
| 7 | 1.40 | 1.40 | 1.25 | 35.97 | 1.87 | 1.72 | 1.98 | 3.12 |
| 8 | 1.40 | 1.40 | 1.25 | 34.68 | 1.86 | 1.69 | **1.98** | **3.25** |
| 9 | 1.42 | 1.43 | 1.26 | 32.65 | 1.92 | 1.73 | 2.08 | 3.17 |

Table 1: *Left: the parameters are free to take on negative values. Right: parameters are constrained through exponentiation so that the resulting function is both positive and monotone increasing everywhere w.r.t. to both inputs. Top: regular feedforward artificial neural networks. Bottom: neural networks with an architecture resembling the Black-Scholes formula as defined in equation 9. The number of units varies from 1 to 9 for each network architecture. The first two quarters of 1988 were used for training, the third of 1988 for validation and the fourth of 1988 for testing. The first quarter of 1989 was used as a second test set to assess the persistence of the models through time (figure 1). In bold: test results for models with best validation results.*

As can be seen in tables 1 and 2, the positivity constraints through exponentiation of the weights allow the networks to avoid overfitting. The training errors are generally slightly lower for the networks with unconstrained weights, the validation errors are similar but final test errors are disastrous for unconstrained networks, compared to the constrained ones. This "liftoff" pattern when looking at training, validation and testing errors has triggered our attention towards the analysis of the evolution of the test error through time. The unconstrained networks obtain better training, validation and testing (test 1) results but fail in

**Products of SoftPlus and Sigmoid Functions**
Mean Squared Error Results on Call Option Pricing ($\times 10^{-4}$)

| Units | Unconstrained weights | | | | Constrained weights | | | |
|-------|-------|-------|-------|-------|-------|-------|-------|-------|
| | Train | Valid | Test1 | Test2 | Train | Valid | Test1 | Test2 |
| 1 | 2.27 | 2.15 | 2.35 | 3.27 | 2.28 | 2.14 | 2.37 | 3.51 |
| 2 | 1.61 | 1.58 | 1.58 | 14.24 | 2.28 | 2.13 | 2.37 | 3.48 |
| 3 | 1.51 | 1.53 | 1.38 | 18.16 | 2.28 | 2.13 | 2.36 | 3.48 |
| 4 | 1.46 | 1.51 | 1.29 | 20.14 | 1.84 | 1.54 | **1.97** | **4.19** |
| 5 | 1.57 | 1.57 | 1.46 | 10.03 | 1.83 | 1.56 | 1.95 | 4.18 |
| 6 | 1.51 | 1.53 | 1.35 | 22.47 | 1.85 | 1.57 | 1.97 | 4.09 |
| 7 | 1.62 | 1.67 | 1.46 | 7.78 | 1.86 | 1.55 | 2.00 | 4.10 |
| 8 | 1.55 | 1.54 | 1.44 | 11.58 | 1.84 | 1.55 | 1.96 | 4.25 |
| 9 | 1.46 | 1.47 | **1.31** | **26.13** | 1.87 | 1.60 | 1.97 | 4.12 |

**Sums of SoftPlus and Sigmoid functions**
Mean Squared Error Results on Call Option Pricing ($\times 10^{-4}$)

| Units | Unconstrained weights | | | | Constrained weights | | | |
|-------|-------|-------|-------|-------|-------|-------|-------|-------|
| | Train | Valid | Test1 | Test2 | Train | Valid | Test1 | Test2 |
| 1 | 1.83 | 1.59 | 1.93 | 4.10 | 2.30 | 2.19 | 2.36 | 3.43 |
| 2 | 1.42 | 1.45 | **1.26** | **25.00** | 2.29 | 2.19 | 2.34 | 3.39 |
| 3 | 1.45 | 1.46 | 1.32 | 35.00 | 1.84 | 1.58 | 1.95 | 4.11 |
| 4 | 1.56 | 1.69 | 1.33 | 21.80 | 1.85 | 1.56 | 1.99 | 4.09 |
| 5 | 1.60 | 1.69 | 1.42 | 10.11 | 1.85 | 1.52 | **2.00** | **4.21** |
| 6 | 1.57 | 1.66 | 1.39 | 14.99 | 1.86 | 1.54 | 2.00 | 4.12 |
| 7 | 1.61 | 1.67 | 1.48 | 8.00 | 1.86 | 1.60 | 1.98 | 3.94 |
| 8 | 1.64 | 1.72 | 1.48 | 7.89 | 1.85 | 1.54 | 1.98 | 4.25 |
| 9 | 1.65 | 1.70 | 1.52 | 6.16 | 1.84 | 1.54 | 1.97 | 4.25 |

Table 2: *Similar results as in table 1 but for two new architectures. Top: products of softplus along the convex axis with sigmoid along the monotone axis. Bottom: the softplus and sigmoid functions are summed instead of being multiplied. Top right: the fully constrained proposed architecture.*

the extra testing set (test 2). Constrained architectures seem more robust to changes in underlying econometric conditions. The constrained Black-Scholes similar model performs slightly better than other models on the second test set but then fails on latter quarters (figure 1). All in all, at the expense of slightly higher initial errors our proposed architecture allows us to forecast with increased stability much farther in the future. This is a very welcome property as new derivative products have a tendency to lock in values for much longer durations (up to 10 years) than traditional ones.

# 6 Conclusions

Motivated by prior knowledge on the derivatives of the function that gives the price of European options, we have introduced new classes of functions similar to multi-layer neural networks that have those properties. We have shown one of these classes to be a universal approximator for functions having those properties, and we have shown that using this *a priori* knowledge can help in improving generalization performance. In particular, we have found that the models that incorporate this *a priori* knowledge generalize in a more stable way over time.

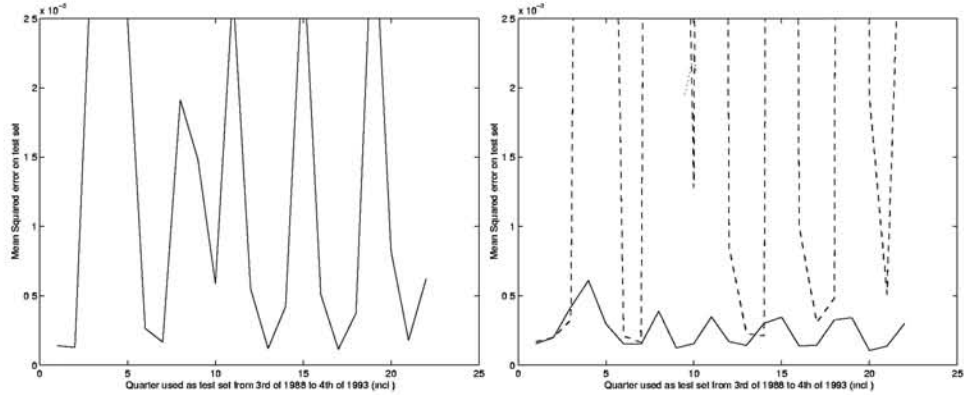

Figure 1: *Out-of-sample results from the third quarter of 1988 to the fourth of 1993 (incl.) for models with best validation results. Left: unconstrained models: results for the Black-Scholes similar network. Other unconstrained models exhibit similar swinging result patterns and levels of errors. Right: constrained models: the fully constrained proposed architecture (solid). The model with sums over dimensions obtains similar results. The regular neural network (dotted). The constrained Black-Scholes model obtains very poor results (dashed).*

## Footnotes

\*C.N. is now with Health Canada at Claude_Nadeau@hc-sc.gc.ca

## References

[1] G. Cybenko. Continuous valued neural networks with two hidden layers are sufficient. Technical report, Department of Computer Science, Tufts University, Medford, MA, 1988.

[2] G. Cybenko. Approximation by superpositions of a sigmoidal function. 2:303–314, 1989.

[3] C. Dugas, O. Bardou, and Y. Bengio. Analyses empiriques sur des transactions d'options. Technical Report 1176, Départment d'informatique et de Recherche Opérationnelle, Université de Montréal, Montréal, Québec, Canada, 2000.

[4] R. Garcia and R. Gençay. Pricing and Hedging Derivative Securities with Neural Networks and a Homogeneity Hint. Technical Report 98s-35, CIRANO, Montréal, Québec, Canada, 1998.

[5] K. Hornik, M. Stinchcombe, and H. White. Multilayer feedforward networks are universal approximators. 2:359–366, 1989.

[6] J. Moody. Prediction risk and architecture selection for neural networks. In *From Statistics to Neural Networks: Theory and Pattern Recognition Applications*. Springer, 1994.
